# Towards a general independent subspace analysis

**Fabian J. Theis**
Max Planck Institute for Dynamics and Self-Organisation &
Bernstein Center for Computational Neuroscience
Bunsenstr. 10, 37073 Göttingen, Germany
`fabian@theis.name`

## Abstract

The increasingly popular independent component analysis (ICA) may only be applied to data following the generative ICA model in order to guarantee algorithm-independent and theoretically valid results. Subspace ICA models generalize the assumption of component independence to independence between groups of components. They are attractive candidates for dimensionality reduction methods, however are currently limited by the assumption of equal group sizes or less general semi-parametric models. By introducing the concept of irreducible independent subspaces or components, we present a generalization to a parameter-free mixture model. Moreover, we relieve the condition of at-most-one-Gaussian by including previous results on non-Gaussian component analysis. After introducing this general model, we discuss joint block diagonalization with unknown block sizes, on which we base a simple extension of JADE to algorithmically perform the subspace analysis. Simulations confirm the feasibility of the algorithm.

## 1 Independent subspace analysis

A random vector $\mathbf{Y}$ is called an *independent component* of the random vector $\mathbf{X}$, if there exists an invertible matrix $\mathbf{A}$ and a decomposition $\mathbf{X} = \mathbf{A}(\mathbf{Y}, \mathbf{Z})$ such that $\mathbf{Y}$ and $\mathbf{Z}$ are stochastically independent. The goal of a general *independent subspace analysis (ISA)* or *multidimensional independent component analysis* is the decomposition of an arbitrary random vector $\mathbf{X}$ into independent components. If $\mathbf{X}$ is to be decomposed into one-dimensional components, this coincides with ordinary independent component analysis (ICA). Similarly, if the independent components are required to be of the same dimension $k$, then this is denoted by multidimensional ICA of fixed group size $k$ or simply $k$-ISA. So 1-ISA is equivalent to ICA.

### 1.1 Why extend ICA?

An important structural aspect in the search for decompositions is the knowledge of the number of solutions i.e. the indeterminacies of the problem. Without it, the result of any ICA or ISA algorithm cannot be compared with other solutions, so for instance blind source separation (BSS) would be impossible. Clearly, given an ISA solution, invertible transforms in each component (scaling matrices $\mathbf{L}$) as well as permutations of components of the same dimension (permutation matrices $\mathbf{P}$) give again an ISA of $\mathbf{X}$. And indeed, in the special case of ICA, scaling and permutation are already all indeterminacies given that at most one Gaussian is contained in $\mathbf{X}$ [6]. This is one of the key theoretical results in ICA, allowing the usage of ICA for solving BSS problems and hence stimulating many applications. It has been shown that also for $k$-ISA, scalings and permutations as above are the only indeterminacies [11], given some additional rather weak restrictions to the model.

However, a serious drawback of $k$-ISA (and hence of ICA) lies in the fact that the requirement fixed group-size $k$ does not allow us to apply this analysis to an arbitrary random vector. Indeed,

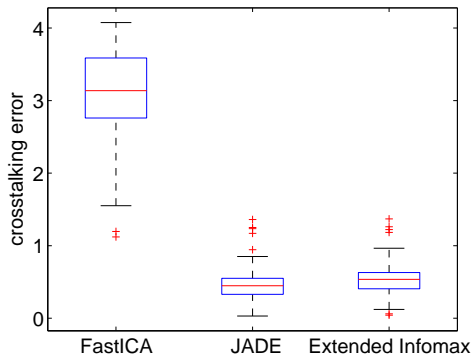

Figure 1: Applying ICA to a random vector $\mathbf{X} = \mathbf{AS}$ that does not fulfill the ICA model; here $\mathbf{S}$ is chosen to consist of a two-dimensional and a one-dimensional irreducible component. Shown are the statistics over 100 runs of the Amari error of the random original and the reconstructed mixing matrix using the three ICA-algorithms FastICA, JADE and Extended Infomax. Clearly, the original mixing matrix could not be reconstructed in any of the experiments. However, interestingly, the latter two algorithms do indeed find an ISA up to permutation, which will be explained in section 3.

theoretically speaking, it may only be applied to random vectors following the $k$-ISA blind source separation model, which means that they have to be mixtures of a random vector that consists of independent groups of size $k$. If this is the case, uniqueness up to permutation and scaling holds as noted above; however if $k$-ISA is applied to any random vector, a decomposition into groups that are only 'as independent as possible' cannot be unique and depends on the contrast and the algorithm. In the literature, ICA is often applied to find representations fulfilling the independence condition as well as possible, however care has to be taken; the strong uniqueness result is not valid any more, and the results may depend on the algorithm as illustrated in figure 1.

This work aims at finding an ISA model that allows applicability to any random vector. After reviewing previous approaches, we will provide such a model together with a corresponding uniqueness result and a preliminary algorithm.

## 1.2 Previous approaches to ISA for dependent component analysis

Generalizations of the ICA model that are to include dependencies of multiple one-dimensional components have been studied for quite some time. ISA in the terminology of multidimensional ICA has first been introduced by Cardoso [4] using geometrical motivations. His model as well as the related but independently proposed factorization of multivariate function classes [9] is quite general, however no identifiability results were presented, and applicability to an arbitrary random vector was unclear; later, in the special case of equal group sizes ($k$-ISA) uniqueness results have been extended from the ICA theory [11]. Algorithmic enhancements in this setting have been recently studied by [10]. Moreover, if the observation contain additional structures such as spatial or temporal structures, these may be used for the multidimensional separation [13].

Hyvärinen and Hoyer presented a special case of $k$-ISA by combining it with invariant feature subspace analysis [7]. They model the dependence within a $k$-tuple explicitly and are therefore able to propose more efficient algorithms without having to resort to the problematic multidimensional density estimation. A related relaxation of the ICA assumption is given by topographic ICA [8], where dependencies between all components are assumed and modelled along a topographic structure (e.g. a 2-dimensional grid). Bach and Jordan [2] formulate ISA as a component clustering problem, which necessitates a model for inter-cluster independence and intra-cluster dependence. For the latter, they propose to use a tree-structure as employed by their tree dependepent component analysis. Together with inter-cluster independence, this implies a search for a transformation of the mixtures into a forest i.e. a set of disjoint trees. However, the above models are all semi-parametric and hence not fully blind. In the following, no additional structures are necessary for the separation.

### 1.3 General ISA

**Definition 1.1.** *A random vector $\mathbf{S}$ is said to be* irreducible *if it contains no lower-dimensional independent component. An invertible matrix $\mathbf{W}$ is called a (general)* independent subspace analysis *of $\mathbf{X}$ if $\mathbf{WX} = (\mathbf{S}_1, \ldots, \mathbf{S}_k)$ with pairwise independent, irreducible random vectors $\mathbf{S}_i$.*

Note that in this case, the $\mathbf{S}_i$ are independent components of $\mathbf{X}$. The idea behind this definition is that in contrast to ICA and $k$-ISA, we do not fix the size of the groups $\mathbf{S}_i$ in advance. Of course, some restriction is necessary, otherwise no decomposition would be enforced at all. This restriction is realized by allowing only irreducible components. The advantage of this formulation now is that it can clearly be applied to any random vector, although of course a trivial decomposition might be the result in the case of an irreducible random vector. Obvious indeterminacies of an ISA of $\mathbf{X}$ are, as mentioned above, scalings i.e. invertible transformations within each $\mathbf{S}_i$ and permutation of $\mathbf{S}_i$ of the same dimension[1]. These are already all indeterminacies as shown by the following theorem, which extends previous results in the case of ICA [6] and $k$-ISA [11], where also the additional slight assumptions on square-integrability i.e. on existing covariance have been made.

**Theorem 1.2.** *Given a random vector $\mathbf{X}$ with existing covariance and no Gaussian independent component, then an ISA of $\mathbf{X}$ exists and is unique except for scaling and permutation.*

Existence holds trivially but uniqueness is not obvious. Due to the limited space, we only give a short sketch of the proof in the following. The uniqueness result can easily be formulated as a subspace extraction problem, and theorem 1.2 follows readily from

**Lemma 1.3.** *Let $\mathbf{S} = (\mathbf{S}_1, \ldots, \mathbf{S}_k)$ be a square-integrable decomposition of $\mathbf{S}$ into irreducible independent components $\mathbf{S}_i$. If $\mathbf{X}$ is an irreducible component of $\mathbf{S}$, then $\mathbf{X} \sim \mathbf{S}_i$ for some $i$.*

Here the equivalence relation $\sim$ denotes equality except for an invertible transformation. The following two lemmata each give a simplification of lemma 1.3 by ordering the components $\mathbf{S}_i$ according to their dimensions. Some care has to be taken when showing that lemma 1.5 implies lemma 1.4.

**Lemma 1.4.** *Let $\mathbf{S}$ and $\mathbf{X}$ be defined as in lemma 1.3. In addition assume that $\dim \mathbf{S}_i = \dim \mathbf{X}$ for $i \leq l$ and $\dim \mathbf{S}_i < \dim \mathbf{X}$ for $i > l$. Then $\mathbf{X} \sim \mathbf{S}_i$ for some $i \leq l$.*

**Lemma 1.5.** *Let $\mathbf{S}$ and $\mathbf{X}$ be defined as in lemma 1.4, and let $l = 1$ and $k = 2$. Then $\mathbf{X} \sim \mathbf{S}_1$.*

In order to prove lemma 1.5 (and hence the theorem), it is sufficient to show the following lemma:

**Lemma 1.6.** *Let $\mathbf{S} = (\mathbf{S}_1, \mathbf{S}_2)$ with $\mathbf{S}_1$ irreducible and $m := \dim \mathbf{S}_1 > \dim \mathbf{S}_2 =: n$. If $\mathbf{X} = \mathbf{AS}$ is again irreducible for some $m \times (m + n)$-matrix $\mathbf{A}$, then (i) the left $m \times m$-submatrix of $\mathbf{A}$ is invertible, and (ii) if $\mathbf{X}$ is an independent component of $\mathbf{S}$, the right $m \times n$-submatrix of $\mathbf{A}$ vanishes.*

(i) follows after some linear algebra, and is necessary to show the more difficult part (ii). For this, we follow the ideas presented in [12] using factorization of the joint characteristic function of $\mathbf{S}$.

### 1.4 Dealing with Gaussians

In the previous section, Gaussians had to be excluded (or at most one was allowed) in order to avoid additional indeterminacies. Indeed, any orthogonal transformation of two decorrelated hence independent Gaussians is again independent, so clearly such a strong identification result would not be possible.

Recently, a general decomposition model dealing with Gaussians was proposed in the form of the so-called *non-Gaussian subspace analysis (NGSA)* [3]. It tries to detect a whole non-Gaussian subspace within the data, and no assumption of independence within the subspace is made. More precisely, given a random vector $\mathbf{X}$, a factorization $\mathbf{X} = \mathbf{AS}$ with an invertible matrix $\mathbf{A}$, $\mathbf{S} = (\mathbf{S}_N, \mathbf{S}_G)$ and $\mathbf{S}_N$ a square-integrable $m$-dimensional random vector is called an $m$-*decomposition* of $\mathbf{X}$ if $\mathbf{S}_N$ and $\mathbf{S}_G$ are stochastically independent and $\mathbf{S}_G$ is Gaussian. In this case, $\mathbf{X}$ is said to be $m$-*decomposable*. $\mathbf{X}$ is denoted to be *minimally $n$-decomposable* if $\mathbf{X}$ is not $(n-1)$-decomposable. According to our previous notation, $\mathbf{S}_N$ and $\mathbf{S}_G$ are independent components of $\mathbf{X}$. It has been shown that the subspaces of such decompositions are unique [12]:

**Theorem 1.7** (Uniqueness of NGSA). *The mixing matrix $\mathbf{A}$ of a minimal decomposition is unique except for transformations in each of the two subspaces.*

Moreover, explicit algorithms can be constructed for identifying the subspaces [3]. This result enables us to generalize theorem 1.2and to get a general decomposition theorem, which characterizes solutions of ISA.

**Theorem 1.8** (Existence and Uniqueness of ISA). *Given a random vector $\mathbf{X}$ with existing covariance, an ISA of $\mathbf{X}$ exists and is unique except for permutation of components of the same dimension and invertible transformations within each independent component and within the Gaussian part.*

*Proof.* Existence is obvious. Uniqueness follows after first applying theorem 1.7 to $\mathbf{X}$ and then theorem 1.2 to the non-Gaussian part. $\square$

## 2   Joint block diagonalization with unknown block-sizes

Joint diagonalization has become an important tool in ICA-based BSS (used for example in JADE) or in BSS relying on second-order temporal decorrelation. The task of (real) *joint diagonalization (JD)* of a set of symmetric real $n \times n$ matrices $\mathcal{M} := \{\mathbf{M}_1, \ldots, \mathbf{M}_K\}$ is to find an orthogonal matrix $\mathbf{E}$ such that $\mathbf{E}^\top \mathbf{M}_k \mathbf{E}$ is diagonal for all $k = 1, \ldots, K$ i.e. to minimize $f(\hat{\mathbf{E}}) := \sum_{k=1}^{K} \|\hat{\mathbf{E}}^\top \mathbf{M}_k \hat{\mathbf{E}} - \mathrm{diagM}(\hat{\mathbf{E}}^\top \mathbf{M}_k \hat{\mathbf{E}})\|_F^2$ with respect to the orthogonal matrix $\hat{\mathbf{E}}$, where $\mathrm{diagM}(\mathbf{M})$ produces a matrix where all off-diagonal elements of $\mathbf{M}$ have been set to zero, and $\|\mathbf{M}\|_F^2 := \mathrm{tr}(\mathbf{MM}^\top)$ denotes the squared Frobenius norm. The Frobenius norm is invariant under conjugation by an orthogonal matrix, so minimizing $f$ is equivalent to maximizing $g(\hat{\mathbf{E}}) := \sum_{k=1}^{K} \|\mathrm{diag}(\hat{\mathbf{E}}^\top \mathbf{M}_k \hat{\mathbf{E}})\|^2$, where now $\mathrm{diag}(\mathbf{M}) := (m_{ii})_i$ denotes the diagonal of $\mathbf{M}$. For the actual minimization of $f$ respectively maximization of $g$, we will use the common approach of Jacobi-like optimization by iterative applications of Givens rotation in two coordinates [5].

### 2.1   Generalization to blocks

In the following we will use a generalization of JD in order to solve ISA problems. Instead of fully diagonalizing all $n \times n$ matrices $\mathbf{M}_k \in \mathcal{M}$, in *joint block diagonalization (JBD)* of $\mathcal{M}$ we want to determine $\mathbf{E}$ such that $\mathbf{E}^\top \mathbf{M}_k \mathbf{E}$ is block-diagonal. Depending on the application, we fix the block-structure in advance or try to determine it from $\mathcal{M}$. We are not interested in the order of the blocks, so the block-structure is uniquely specified by fixing a *partition of $n$* i.e. a way of writing $n$ as a sum of positive integers, where the order of the addends is not significant. So let[2] $n = m_1 + \ldots + m_r$ with $m_1 \leq m_2 \leq \ldots \leq m_r$ and set $\mathbf{m} := (m_1, \ldots, m_r) \in \mathbb{N}^r$. An $n \times n$ matrix is said to be $\mathbf{m}$-*block diagonal* if it is of the form

$$\begin{pmatrix} \mathbf{D}_1 & \cdots & 0 \\ \vdots & \ddots & \vdots \\ 0 & \cdots & \mathbf{D}_r \end{pmatrix}$$

with arbitrary $m_i \times m_i$ matrices $\mathbf{D}_i$.

As generalization of JD in the case of known the block structure, we can formulate the *joint $\mathbf{m}$-block diagonalization ($\mathbf{m}$-JBD)* problem as the minimization of $f^{\mathbf{m}}(\hat{\mathbf{E}}) := \sum_{k=1}^{K} \|\hat{\mathbf{E}}^\top \mathbf{M}_k \hat{\mathbf{E}} - \mathrm{diagM}^{\mathbf{m}}(\hat{\mathbf{E}}^\top \mathbf{M}_k \hat{\mathbf{E}})\|_F^2$ with respect to the orthogonal matrix $\hat{\mathbf{E}}$, where $\mathrm{diagM}^{\mathbf{m}}(\mathbf{M})$ produces a $\mathbf{m}$-block diagonal matrix by setting all other elements of $\mathbf{M}$ to zero. In practice due to estimation errors, such $\mathbf{E}$ will not exist, so we speak of approximate JBD and imply minimizing some error-measure on non-block-diagonality. Indeterminacies of any $\mathbf{m}$-JBD are $\mathbf{m}$-*scaling* i.e. multiplication by an $\mathbf{m}$-block diagonal matrix from the right, and $\mathbf{m}$-*permutation* defined by a permutation matrix that only swaps blocks of the same size.

Finally, we speak of *general JBD* if we search for a JBD but no block structure is given; instead it is to be determined from the matrix set. For this it is necessary to require a block

structure of maximal length, otherwise trivial solutions or 'in-between' solutions could exist (and obviously contain high indeterminacies). Formally, $\mathbf{E}$ is said to be a (general) JBD of $\mathcal{M}$ if $(\mathbf{E}, \mathbf{m}) = \text{argmax}_{\mathbf{m} \mid \exists \mathbf{E}: f^{\mathbf{m}}(\mathbf{E})=0} |\mathbf{m}|$. In practice due to errors, a true JBD would always result in the trivial decomposition $\mathbf{m} = (n)$, so we define an approximate general JBD by requiring $f^{\mathbf{m}}(\mathbf{E}) < \epsilon$ for some fixed constant $\epsilon > 0$ instead of $f^{\mathbf{m}}(\mathbf{E}) = 0$.

## 2.2 JBD by JD

A few algorithms to actually perform JBD have been proposed, see [1] and references therein. In the following we will simply perform joint diagonalization and then permute the columns of $\mathbf{E}$ to achieve block-diagonality — in experiments this turns out to be an efficient solution to JBD [1]. This idea has been formulated in a conjecture [1] essentially claiming that a minimum of the JD cost function $f$ already is a JBD i.e. a minimum of the function $f^{\mathbf{m}}$ up to a permutation matrix. Indeed, in the conjecture it is required to use the Jacobi-update algorithm from [5], but this is not necessary, and we can prove the conjecture partially:

We want to show that JD implies JBD up to permutation, i.e. if $\mathbf{E}$ is a minimum of $f$, then there exists a permutation $\mathbf{P}$ such that $f^{\mathbf{m}}(\mathbf{EP}) = 0$ (given existence of a JBD of $\mathcal{M}$). But of course $f(\mathbf{EP}) = f(\mathbf{E})$, so we will show why (certain) JBD solutions are minima of $f$. However, JD might have additional minima. First note that clearly not any JBD minimizes $f$, only those such that in each block of size $m_k$, $f(\mathbf{E})$ when restricted to the block is maximal over $\mathbf{E} \in O(m_k)$. We will call such a JBD *block-optimal* in the following.

**Theorem 2.1.** *Any block-optimal JBD of $\mathcal{M}$ (zero of $f^{\mathbf{m}}$) is a local minimum of $f$.*

*Proof.* Let $\mathbf{E} \in O(n)$ be block-optimal with $f^{\mathbf{m}}(\mathbf{E}) = 0$. We have to show that $\mathbf{E}$ is a local minimum of $f$ or equivalently a local maximum of the squared diagonal sum $g$. After substituting each $\mathbf{M}_k$ by $\mathbf{E}^\top \mathbf{M}_k \mathbf{E}$, we may already assume that $\mathbf{M}_k$ is $\mathbf{m}$-block diagonal, so we have to show that $\mathbf{E} = \mathbf{I}$ is a local maximum of $g$.

Consider the elementary Givens rotation $\mathbf{G}_{ij}(\epsilon)$ defined for $i < j$ and $\epsilon \in (-1, 1)$ as the orthogonal matrix, where all diagonal elements are 1 except for the two elements $\sqrt{1 - \epsilon^2}$ in rows $i$ and $j$ and with all off-diagonal elements equal to 0 except for the two elements $\epsilon$ and $-\epsilon$ at $(i, j)$ and $(j, i)$, respectively. It can be used to construct local coordinates of the $d := n(n-1)/2$-dimensional manifold $O(n)$ at $\mathbf{I}$, simply by $\iota(\epsilon_{12}, \epsilon_{13}, \ldots, \epsilon_{n-1,n}) := \prod_{i<j} \mathbf{G}_{ij}(\epsilon_{ij})$ This is an embedding, and $\iota(0) = \mathbf{I}$, so we only have to show that $h(\boldsymbol{\epsilon}) := g(\iota(\boldsymbol{\epsilon}))$ has a local maximum at $\boldsymbol{\epsilon} = 0$. We do this by considering $h$ partially in each coordinate. Let $i < j$. If $i, j$ are in the same block of $\mathbf{m}$, then $h$ is locally maximal i.e. negative semi-definite at 0 in the direction $\epsilon_{ij}$ because of block-optimality.

Now assume $i$ and $j$ are from different blocks. After possible permutation, we may assume that $j = i + 1$ so that each matrix $\mathbf{M}_k \in \mathcal{M}$ has $(M_k)_{ij} = (M_k)_{ji} = 0$, and $a_k := (M_k)_{ii}, b_k := (M_k)_{jj}$. Then $\mathbf{G}_{ij}(\epsilon)^\top \mathbf{M}_k \mathbf{G}_{ij}(\epsilon)$ can be easily calculated at coordinates $(i, i)$ to $(j, j)$, and indeed entries on the diagonal other than at indices $(i, i)$ and $(j, j)$ are not changed, so

$$\|\text{diag}(\mathbf{G}_{ij}(\epsilon)^\top \mathbf{M}_k \mathbf{G}_{ij}(\epsilon))\|^2 - \|\text{diag}(\mathbf{M}_k)\|^2 =$$
$$= -2a_k(a_k - b_k)\epsilon^2 + 2b_k(a_k - b_k)\epsilon^2 + 2(a_k - b_k)^2\epsilon^4$$
$$= -2(a_k^2 + b_k^2)\epsilon^2 + 2(a_k - b_k)^2\epsilon^4.$$

Hence $h(0, \ldots, 0, \epsilon_{ij}, 0, \ldots, 0) - h(0) = -c\epsilon_{ij}^2 + d\epsilon_{ij}^4$ with $c = 2\sum_{k=1}^{K}(a_k^2 + b_k^2)$ and $d = 2\sum_{k=1}^{K}(a_k - b_k)^2$. Now either $c = 0$, then also $d = 0$ and $h$ is constant zero in the direction $\epsilon_{ij}$. Or, more interestingly, $c \neq 0$, then $c > 0$ and therefore $h$ is negative definite in the direction $\epsilon_{ij}$. Altogether we get a negative definite $h$ at 0 except for 'trivial directions', and hence a local maximum at 0. $\square$

## 2.3 Recovering the permutation

In order to perform JBD, we therefore only have to find a JD $\mathbf{E}$ of $\mathcal{M}$. What is left according to the above theorem is to find a permutation matrix $\mathbf{P}$ such that $\mathbf{EP}$ block-diagonalizes $\mathcal{M}$. In the case of known block-order $\mathbf{m}$, we can employ similar techniques as used in [1, 10], which essentially find $\mathbf{P}$ by some combinatorial optimization.

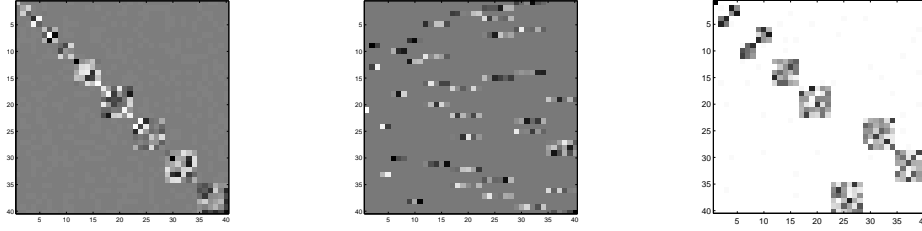

(a) (unknown) block diagonal $\mathbf{M}_1$ (b) $\hat{\mathbf{E}}^\top \mathbf{E}$ w/o recovered permutation (c) $\hat{\mathbf{E}}^\top \mathbf{E}$

Figure 2: Performance of the proposed general JBD algorithm in the case of the (unknown) block-partition $40 = 1 + 2 + 2 + 3 + 3 + 5 + 6 + 6 + 6 + 6$ in the presence of noise with SNR of $5dB$. The product $\hat{\mathbf{E}}^\top \mathbf{E}$ of the inverse of the estimated block diagonalizer and the original one is an $\mathbf{m}$-block diagonal matrix except for permutation within groups of the same sizes as claimed in section 2.2.

In the case of unknown block-size, we propose to use the following simple permutation-recovery algorithm: consider the mean diagonalized matrix $\overline{\mathbf{D}} := K^{-1} \sum_{k=1}^{K} \mathbf{E}^\top \mathbf{M}_k \mathbf{E}$. Due to the assumption that $\mathcal{M}$ is $\mathbf{m}$-block-diagonalizable (with unknown $\mathbf{m}$), each $\mathbf{E}^\top \mathbf{M}_k \mathbf{E}$ and hence also $\overline{\mathbf{D}}$ must be $\mathbf{m}$-block-diagonal except for a permutation $\mathbf{P}$, so it must have the corresponding number of zeros in each column and row. In the approximate JBD case, thresholding with a threshold $\theta$ is necessary, whose choice is non-trivial.

We propose using algorithm 1 to recover the permutation; we denote its resulting permuted matrix by $\mathcal{P}(\mathbf{D})$ when applied to the input $\mathbf{D}$. $\mathcal{P}(\overline{\mathbf{D}})$ is constructed from possibly thresholded $\overline{\mathbf{D}}$ by iteratively permuting columns and rows in order to guarantee that all non-zeros of $\overline{\mathbf{D}}$ are clustered along the diagonal as closely as possible. This recovers the permutation as well as the partition $\mathbf{m}$ of $n$.

---
**Algorithm 1**: Block-diagonality permutation finder
---
**Input**: $(n \times n)$-matrix $\mathbf{D}$
**Output**: block-diagonal matrix $\mathcal{P}(\mathbf{D}) := \mathbf{D}'$ such that $\mathbf{D}' = \mathbf{P}\mathbf{D}\mathbf{P}^T$ for a permutation matrix $\mathbf{P}$

$\mathbf{D}' \leftarrow \mathbf{D}$
**for** $i \leftarrow 1$ **to** $n$ **do**
    **repeat**
        **if** $(j_0 \leftarrow \min\{j|j \geq i \text{ and } d'_{ij} = 0 \text{ and } d'_{ji} = 0\})$ *exists* **then**
            **if** $(k_0 \leftarrow \min\{k|k > j_0 \text{ and } (d'_{ik} \neq 0 \text{ or } d'_{ki} \neq 0)\})$ *exists* **then**
                swap column $j_0$ of $\mathbf{D}'$ with column $k_0$
                swap row $j_0$ of $\mathbf{D}'$ with row $k_0$
    **until** *no swap has occurred* ;
---

We illustrate the performance of the proposed JBD algorithm as follows: we generate a set of $K = 100$ $\mathbf{m}$-block-diagonal matrices $\mathbf{D}_k$ of dimension $40 \times 40$ with $\mathbf{m} = (1, 2, 2, 3, 3, 5, 6, 6, 6, 6)$. They have been generated in blocks of size $\mathbf{m}$ with coefficients chosen randomly uniform from $[-1, 1]$, and symmetrized by $\mathbf{D}_k \leftarrow (\mathbf{D}_k + \mathbf{D}_k^\top)/2$. After that, they have been mixed by a random orthogonal mixing matrix $\mathbf{E} \in O(40)$, i.e. $\mathbf{M}_k := \mathbf{E}\mathbf{D}_k\mathbf{E}^\top + \mathbf{N}$, where $\mathbf{N}$ is a noise matrix with independent Gaussian entries such that the resulting signal-to-noise ratio is $5dB$. Application of the JBD algorithm from above to $\{\mathbf{M}_1, \ldots, \mathbf{M}_K\}$ with threshold $\theta = 0.1$ correctly recovers the block sizes, and the estimated block diagonalizer $\hat{\mathbf{E}}$ equals $\mathbf{E}$ up to $\mathbf{m}$-scaling and permutation, as illustrated in figure 2.

## 3 SJADE — a simple algorithm for general ISA

As usual by preprocessing of the observations $\mathbf{X}$ by whitening we may assume that $\text{Cov}(\mathbf{X}) = \mathbf{I}$. The indeterminacies allow scaling transformations in the sources, so without loss of generality let

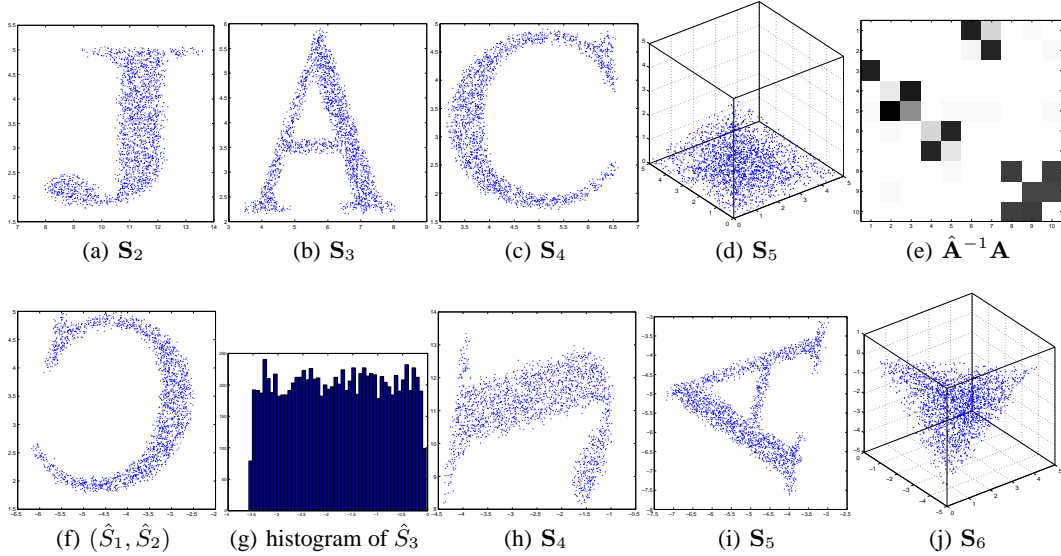

(a) $\mathbf{S}_2$     (b) $\mathbf{S}_3$     (c) $\mathbf{S}_4$     (d) $\mathbf{S}_5$     (e) $\hat{\mathbf{A}}^{-1}\mathbf{A}$

(f) $(\hat{S}_1, \hat{S}_2)$     (g) histogram of $\hat{S}_3$     (h) $\mathbf{S}_4$     (i) $\mathbf{S}_5$     (j) $\mathbf{S}_6$

Figure 3: Example application of general ISA for unknown sizes $\mathbf{m} = (1, 2, 2, 2, 3)$. Shown are the scatter plots i.e. densities of the source components and the mixing-separating map $\hat{\mathbf{A}}^{-1}\mathbf{A}$.

also $\mathrm{Cov}(\mathbf{S}) = \mathbf{I}$. Then $\mathbf{I} = \mathrm{Cov}(\mathbf{X}) = \mathbf{A}\,\mathrm{Cov}(\mathbf{S})\mathbf{A}^\top = \mathbf{A}\mathbf{A}^\top$ so $\mathbf{A}$ is orthogonal. Due to the ISA assumptions, the fourth-order cross cumulants of the sources have to be trivial between different groups, and within the Gaussians. In order to find transformations of the mixtures fulfilling this property, we follow the idea of the JADE algorithmbut now in the ISA setting. We perform JBD of the (whitened) *contracted quadricovariance matrices* defined by $\mathbf{C}_{ij}(\mathbf{X}) := E\left(\mathbf{X}^\top \mathbf{E}_{ij}\mathbf{X}\mathbf{X}\mathbf{X}^\top\right) - \mathbf{E}_{ij} - \mathbf{E}_{ij}^\top - \mathrm{tr}(\mathbf{E}_{ij})\mathbf{I}$. Here $\mathbf{R}_{\mathbf{X}} := \mathrm{Cov}(\mathbf{X})$ and $\mathbf{E}_{ij}$ is a set of eigen-matrices of $\mathbf{C}_{ij}$, $1 \leq i, j \leq n$. One simple choice is to use $n^2$ matrices $\mathbf{E}_{ij}$ with zeros everywhere except 1 at index $(i, j)$. More elaborate choices of eigen-matrices (with only $n(n+1)/2$ or even $n$ entries) are possible. The resulting algorithm, *subspace-JADE (SJADE)* not only performs NGCA by grouping Gaussians as one-dimensional components with trivial $\mathbf{C}_{ii}$'s, but also automatically finds the subspace partition $\mathbf{m}$ using the general JBD algorithm from section 2.3.

## 4 Experimental results

In a first example, we consider a general ISA problem in dimension $n = 10$ with the unknown partition $\mathbf{m} = (1, 2, 2, 2, 3)$. In order to generate 2- and 3-dimensional irreducible random vectors, we decided to follow the nice visual ideas from [10] and to draw samples from a density following a known shape — in our case 2d-letters or 3d-geometrical shapes. The chosen sources densities are shown in figure 3(a-d). Another 1-dimensional source following a uniform distribution was constructed. Altogether $10^4$ samples were used. The sources $\mathbf{S}$ were mixed by a mixing matrix $\mathbf{A}$ with coefficients uniformly randomly sampled from $[-1, 1]$ to give mixtures $\mathbf{X} = \mathbf{A}\mathbf{S}$. The recovered mixing matrix $\hat{\mathbf{A}}$ was then estimated using the above block-JADE algorithm with unknown block size; we observed that the method is quite sensitive to the choice of the threshold (here $\theta = 0.015$). Figure 3(e) shows the composed mixing-separating system $\hat{\mathbf{A}}^{-1}\mathbf{A}$; clearly the matrices are equal except for block permutation and scaling, which experimentally confirms theorem 1.8. The algorithm found a partition $\hat{\mathbf{m}} = (1, 1, 1, 2, 2, 3)$, so one 2d-source was misinterpreted as two 1d-sources, but by using previous knowledge combination of the correct two 1d-sources yields the original 2d-source. The resulting recovered sources $\hat{\mathbf{S}} := \hat{\mathbf{A}}^{-1}\mathbf{X}$, figures 3(f-j), then equal the original sources except for permutation and scaling within the sources — which in the higher-dimensional cases implies transformations such as rotation of the underlying images or shapes. When applying ICA (1-ISA) to the above mixtures, we cannot expect to recover the original sources as explained in figure 1; however, some algorithms might recover the sources up to permutation. Indeed, SJADE equals JADE with additional permutation recovery because the joint block diagonalization is per-

formed using joint diagonalization. This explains why JADE retrieves meaningful components even in this non-ICA setting as observed in [4].

In a second example, we illustrate how the algorithm deals with Gaussian sources i.e. how the subspace JADE also includes NGCA. For this we consider the case $n = 5$, $\mathbf{m} = (1, 1, 1, 2)$ and sources with two Gaussians, one uniform and a 2-dimensional irreducible component as before; $10^5$ samples were drawn. We perform 100 Monte-Carlo simulations with random mixing matrix $\mathbf{A}$, and apply SJADE with $\theta = 0.01$. The recovered mixing matrix $\hat{\mathbf{A}}$ is compared with $\mathbf{A}$ by taking the ad-hoc measure $\iota(\mathbf{P}) := \sum_{i=1}^{3} \sum_{j=1}^{2} (p_{ij}^2 + p_{ji}^2)$ for $\mathbf{P} := \hat{\mathbf{A}}^{-1} \mathbf{A}$. Indeed, we get nearly perfect recovery in 99 out of 100 runs, the median of $\iota(\mathbf{P})$ is very low with $0.0083$. A single run diverges with $\iota(P) = 3.48$. In order to show that the algorithm really separates the Gaussian part from the other components, we compare the recovered source kurtoses. The median kurtoses are $-0.0006 \pm 0.02$, $-0.003 \pm 0.3$, $-1.2 \pm 0.3$, $-1.2 \pm 0.2$ and $-1.6 \pm 0.2$. The first two components have kurtoses close to zero, so they are the two Gaussians, whereas the third component has kurtosis of around $-1.2$, which equals the kurtosis of a uniform density. This confirms the applicability of the algorithm in the general, noisy ISA setting.

## 5   Conclusion

Previous approaches for independent subspace analysis were restricted either to fixed group sizes or semi-parametric models. In neither case, general applicability to any kind of mixture data set was guaranteed, so blind source separation might fail. In the present contribution we introduce the concept of irreducible independent components and give an identifiability result for this general, parameter-free model together with a novel arbitrary-subspace-size algorithm based on joint block diagonalization. As in ICA, the main uniqueness theorem is an asymptotic result (but includes noisy case via NGCA). However in practice in the finite sample case, due to estimation errors the general joint block diagonality only approximately holds. Our simple solution in this contribution was to choose appropriate thresholds. But this choice is non-trivial, and adaptive methods are to be developed in future works.

## Footnotes

[1] Note that scaling here implies a basis change in the component $\mathbf{S}_i$, so for example in the case of a two-dimensional source component, this might be rotation and sheering. In the example later in figure 3, these indeterminacies can easily be seen by comparing true and estimated sources.

[2]We do not use the convention from Ferrers graphs of specifying partitions in decreasing order, as a visualization of increasing block-sizes seems to be preferable in our setting.

## References

[1] K. Abed-Meraim and A. Belouchrani. Algorithms for joint block diagonalization. In *Proc. EUSIPCO 2004*, pages 209–212, Vienna, Austria, 2004.

[2] F.R. Bach and M.I. Jordan. Finding clusters in independent component analysis. In *Proc. ICA 2003*, pages 891–896, 2003.

[3] G. Blanchard, M. Kawanabe, M. Sugiyama, V. Spokoiny, and K.-R. Müller. In search of non-gaussian components of a high-dimensional distribution. *JMLR*, 7:247–282, 2006.

[4] J.F. Cardoso. Multidimensional independent component analysis. In *Proc. of ICASSP '98*, Seattle, 1998.

[5] J.F. Cardoso and A. Souloumiac. Jacobi angles for simultaneous diagonalization. *SIAM J. Mat. Anal. Appl.*, 17(1):161–164, January 1995.

[6] P. Comon. Independent component analysis - a new concept? *Signal Processing*, 36:287–314, 1994.

[7] A. Hyvärinen and P.O. Hoyer. Emergence of phase and shift invariant features by decomposition of natural images into independent feature subspaces. *Neural Computation*, 12(7):1705–1720, 2000.

[8] A. Hyvärinen, P.O. Hoyer, and M. Inki. Topographic independent component analysis. *Neural Computation*, 13(7):1525–1558, 2001.

[9] J.K. Lin. Factorizing multivariate function classes. In *Advances in Neural Information Processing Systems*, volume 10, pages 563–569, 1998.

[10] B. Poczos and A. Lörincz. Independent subspace analysis using k-nearest neighborhood distances. In *Proc. ICANN 2005*, volume 3696 of *LNCS*, pages 163–168, Warsaw, Poland, 2005. Springer.

[11] F.J. Theis. Uniqueness of complex and multidimensional independent component analysis. *Signal Processing*, 84(5):951–956, 2004.

[12] F.J. Theis and M. Kawanabe. Uniqueness of non-gaussian subspace analysis. In *Proc. ICA 2006*, pages 917–925, Charleston, USA, 2006.

[13] R. Vollgraf and K. Obermayer. Multi-dimensional ICA to separate correlated sources. In *Proc. NIPS 2001*, pages 993–1000, 2001.
